# Learning a Distance Metric from a Network

**Blake Shaw**[*]
Computer Science Dept.
Columbia University
blake@cs.columbia.edu

**Bert Huang**[*]
Computer Science Dept.
Columbia University
bert@cs.columbia.edu

**Tony Jebara**
Computer Science Dept.
Columbia University
jebara@cs.columbia.edu

## Abstract

Many real-world networks are described by both connectivity information and features for every node. To better model and understand these networks, we present *structure preserving metric learning* (SPML), an algorithm for learning a Mahalanobis distance metric from a network such that the learned distances are tied to the inherent connectivity structure of the network. Like the graph embedding algorithm *structure preserving embedding*, SPML learns a metric which is structure preserving, meaning a connectivity algorithm such as $k$-nearest neighbors will yield the correct connectivity when applied using the distances from the learned metric. We show a variety of synthetic and real-world experiments where SPML predicts link patterns from node features more accurately than standard techniques. We further demonstrate a method for optimizing SPML based on stochastic gradient descent which removes the running-time dependency on the size of the network and allows the method to easily scale to networks of thousands of nodes and millions of edges.

## 1 Introduction

The proliferation of social networks on the web has spurred many significant advances in modeling networks [1, 2, 4, 12, 13, 15, 16, 26]. However, while many efforts have been focused on modeling networks as weighted or unweighted graphs [17], or constructing features from links to describe the nodes in a network [14, 25], few techniques have focused on real-world network data which consists of both node features in addition to connectivity information. Many social networks are of this form; on services such as Facebook, Twitter, or LinkedIn, there are profiles which describe each person, as well as the connections they make. The relationship between a node's features and connections is often not explicit. For example, people "friend" each other on Facebook for a variety of reasons: perhaps they share similar parts of their profile such as their school or major, or perhaps they have completely different profiles. We want to learn the relationship between profiles and links from massive social networks such that we can better predict who is likely to connect. To model this relationship, one could simply model each link independently, where one simply learns what characteristics of two profiles imply a possible link. However, this approach completely ignores the structural characteristics of the links in the network. We posit that modeling independent links is insufficient, and in order to better model these networks one must account for the inherent topology of the network as well as the interactions between the features of nodes. We thus propose *structure preserving metric learning* (SPML), a method for learning a distance metric between nodes that preserves the structural network behavior seen in data.

### 1.1 Background

Metric learning algorithms have been successfully applied to many supervised learning tasks such as classification [3, 23, 24]. These methods first build a $k$-nearest neighbors ($k$NN) graph from

---

[*]Blake Shaw is currently at Foursquare, and Bert Huang is currently at the University of Maryland.

training data with a fixed $k$, and then learn a Mahalanobis distance metric which tries to keep connected points with similar labels close while pushing away class impostors, pairs of points which are connected but of different classes. Fundamentally, these supervised methods aim to learn a distance metric such that applying a connectivity algorithm (for instance, $k$-nearest neighbors) under the metric will produce a graph where no point is connected to others with different class labels. In practice, these constraints are enforced with slack. Once the metric is learned, the class label for an unseen datapoint can be predicted by the majority vote of nearby points under the learned metric.

Unfortunately, these metric learning algorithms are not easily applied when we are given a network as input instead of class labels for each point. Under this new regime, we want to learn a metric such that points connected in the network are close and points which are unconnected are more distant. Intuitively, certain features or groups of features should influence how nodes connect, and thus it should be possible to learn a mapping from features to connectivity such that the mapping respects the underlying topological structure of the network. Like previous metric learning methods, SPML learns a metric which reconciles the input features with some auxiliary information such as class labels. In this case, instead of pushing away class impostors, SPML pushes away *graph impostors*, points which are close in terms of distance but which should remain unconnected in order to preserve the topology of the network. Thus SPML learns a metric where the learned distances are inherently tied to the original input connectivity.

Preserving graph topology is possible by enforcing simple linear constraints on distances between nodes [21]. By adapting the constraints from the graph embedding technique *structure preserving embedding*, we formulate simple linear structure preserving constraints for metric learning that enforce that neighbors of each node are closer than all others. Furthermore, we adapt these constraints for an online setting similar to PEGASOS [20] and OASIS [3], such that we can apply SPML to large networks by optimizing with stochastic gradient descent (SGD).

## 2   Structure preserving metric learning

Given as input an adjacency matrix $\mathbf{A} \in \mathbb{B}^{n \times n}$, and node features $\mathbf{X} \in \mathbb{R}^{d \times n}$, *structure preserving metric learning* (SPML) learns a Mahalanobis distance metric parameterized by a positive semidefinite (PSD) matrix $\mathbf{M} \in \mathbb{R}^{d \times d}$, where $\mathbf{M} \succeq 0$. The distance between two points under the metric is defined as $D_{\mathbf{M}}(\mathbf{x}_i, \mathbf{x}_j) = (\mathbf{x}_i - \mathbf{x}_j)^\top \mathbf{M} (\mathbf{x}_i - \mathbf{x}_j)$. When the metric is the identity $\mathbf{M} = I_d$, $D_{\mathbf{M}}(\mathbf{x}_i, \mathbf{x}_j)$ represents the squared Euclidean distance between the $i$'th and $j$'th points. Learning $\mathbf{M}$ is equivalent to learning a linear scaling on the input features $\mathbf{L}\mathbf{X}$ where $\mathbf{M} = \mathbf{L}^\top \mathbf{L}$ and $\mathbf{L} \in \mathbb{R}^{d \times d}$. SPML learns an $\mathbf{M}$ which is *structure preserving*, as defined in Definition 1. Given a *connectivity algorithm* $\mathcal{G}$, SPML learns a metric such that applying $\mathcal{G}$ to the input data using the learned metric produces the input adjacency matrix exactly.[1] Possible choices for $\mathcal{G}$ include maximum weight $b$-matching, $k$-nearest neighbors, $\epsilon$-neighborhoods, or maximum weight spanning tree.

**Definition 1** *Given a graph with adjacency matrix* $\mathbf{A}$*, a distance metric parametrized by* $\mathbf{M} \in \mathbb{R}^{d \times d}$ *is **structure preserving** with respect to a connectivity algorithm* $\mathcal{G}$*, if* $\mathcal{G}(\mathbf{X}, \mathbf{M}) = \mathbf{A}$*.*

### 2.1   Preserving graph topology with linear constraints

To preserve graph topology, we use the same linear constraints as *structure preserving embedding* (SPE) [21], but apply them to $\mathbf{M}$, which parameterizes the distances between points. A useful tool for defining distances as linear constraints on $\mathbf{M}$ is the transformation

$$D_{\mathbf{M}}(\mathbf{x}_i, \mathbf{x}_j) = \mathbf{x}_i^\top \mathbf{M} \mathbf{x}_i + \mathbf{x}_j^\top \mathbf{M} \mathbf{x}_j - \mathbf{x}_i^\top \mathbf{M} \mathbf{x}_j - \mathbf{x}_j^\top \mathbf{M} \mathbf{x}_i, \tag{1}$$

which allows linear constraints on the distances to be written as linear constraints on the $\mathbf{M}$ matrix. For different connectivity schemes below, we present linear constraints which enforce graph structure to be preserved.

**Nearest neighbor graphs**   The $k$-*nearest neighbor algorithm* ($k$-nn) connects each node to the $k$ neighbors to which the node has shortest distance, where $k$ is an input parameter; therefore, setting $k$

to the true degree for each node, the distances to all disconnected nodes must be larger than the distance to the farthest connected neighbor: $D_{\mathbf{M}}(\mathbf{x}_i, \mathbf{x}_j) > (1 - A_{ij}) \max_l(A_{il} D_{\mathbf{M}}(\mathbf{x}_i, \mathbf{x}_l)), \forall i, j$. Similarly, preserving an $\epsilon$-neighborhood graph obeys linear constraints on $\mathbf{M}$: $D_{\mathbf{M}}(\mathbf{x}_i, \mathbf{x}_j) \leq \epsilon, \forall\{i, j | A_{ij} = 1\}$, and $D_{\mathbf{M}}(\mathbf{x}_i, \mathbf{x}_j) \geq \epsilon, \forall\{i, j | A_{ij} = 0\}$. If for each node the connected distances are less than the unconnected distances (or some $\epsilon$), i.e., the metric obeys the above linear constraints, Definition 1 is satisfied, and thus the connectivity computed under the learned metric $\mathbf{M}$ is exactly $\mathbf{A}$.

**Maximum weight subgraphs**   Unlike nearest neighbor algorithms, which select edges greedily for each node, maximum weight subgraph algorithms select edges from a weighted graph to produce a subgraph which has total maximal weight [6]. Given a metric parametrized by $\mathbf{M}$, let the weight between two points $(i, j)$ be the negated pairwise distance between them: $Z_{ij} = -D_{\mathbf{M}}(\mathbf{x}_i, \mathbf{x}_j) = -(\mathbf{x}_i - \mathbf{x}_j)^\top \mathbf{M}(\mathbf{x}_i - \mathbf{x}_j)$. For example, *maximum weight b-matching* finds the maximum weight subgraph while also enforcing that every node has a fixed degree $b_i$ for each $i$'th node. The formulation for *maximum weight spanning tree* is similar. Unfortunately, preserving structure for these algorithms requires enforcing many linear constraints of the form: $\mathrm{tr}(\mathbf{Z}^\top \mathbf{A}) \geq \mathrm{tr}(\mathbf{Z}^\top \tilde{\mathbf{A}}), \forall \tilde{\mathbf{A}} \in \mathcal{G}$. This reveals one critical difference between structure preserving constraints of these algorithms and those of nearest-neighbor graphs: there are exponentially many linear constraints. To avoid an exponential enumeration, the most violated inequalities can be introduced sequentially using a cutting-plane approach as shown in the next section.

## 2.2   Algorithm derivation

By combining the linear constraints from the previous section with a Frobenius norm (denoted $||\cdot||_\mathrm{F}$) regularizer on $\mathbf{M}$ and regularization parameter $\lambda$, we have a simple semidefinite program (SDP) which learns an $\mathbf{M}$ that is structure preserving and has minimal complexity. Algorithm 1 summarizes the naive implementation of SPML when the connectivity algorithm is $k$-nearest neighbors, which is optimized by a standard SDP solver. For maximum weight subgraph connectivity (e.g., $b$-matching), we use a *cutting-plane* method [10], iteratively finding the worst violating constraint and adding it to a working-set. We can find the most violated constraint at each iteration by computing the adjacency matrix $\tilde{\mathbf{A}}$ that maximizes $\mathrm{tr}(\tilde{\mathbf{Z}} \tilde{\mathbf{A}})$ s.t. $\tilde{\mathbf{A}} \in \mathcal{G}$, which can be done using various methods [6, 7, 8]. Each added constraint enforces that the total weight along the edges of the true graph is greater than total weight of any other graph by some margin. Algorithm 2 shows the steps for SPML with cutting-plane constraints.

---

**Algorithm 1** Structure preserving metric learning with nearest neighbor constraints

---

**Input:** $\mathbf{A} \in \mathbb{B}^{n \times n}, \mathbf{X} \in \mathbb{R}^{d \times n}$, and parameter $\lambda$
 1: $\mathcal{K} = \{\mathbf{M} \succeq 0, D_{\mathbf{M}}(\mathbf{x}_i, \mathbf{x}_j) \geq (1 - A_{ij}) \max_l(A_{il} D_{\mathbf{M}}(\mathbf{x}_i, \mathbf{x}_l)) + 1 - \xi \quad \forall_{i,j}\}$
 2: $\tilde{\mathbf{M}} \leftarrow \mathrm{argmin}_{\mathbf{M} \in \mathcal{K}} \frac{\lambda}{2} ||\mathbf{M}||_\mathrm{F}^2 + \xi$ {Found via SDP}
 3: **return** $\tilde{\mathbf{M}}$

---

---

**Algorithm 2** Structure preserving metric learning with cutting-plane constraints

---

**Input:** $\mathbf{A} \in \mathbb{B}^{n \times n}, \mathbf{X} \in \mathbb{R}^{d \times n}$, connectivity algorithm $\mathcal{G}$, and parameters $\lambda, \kappa$
 1: $\mathcal{K} = \{\mathbf{M} \succeq 0\}$
 2: **repeat**
 3:    $\tilde{\mathbf{M}} \leftarrow \mathrm{argmin}_{\mathbf{M} \in \mathcal{K}} \frac{\lambda}{2} ||\mathbf{M}||_\mathrm{F}^2 + \xi$ {Found via SDP}
 4:    $\tilde{\mathbf{Z}} \leftarrow 2\mathbf{X}^\top \tilde{\mathbf{M}} \mathbf{X} - \mathrm{diag}(\mathbf{X}^\top \tilde{\mathbf{M}} \mathbf{X})\mathbf{1}^\top - \mathbf{1}\mathrm{diag}(\mathbf{X}^\top \tilde{\mathbf{M}} \mathbf{X})^\top$
 5:    $\tilde{\mathbf{A}} \leftarrow \mathrm{argmax}_{\tilde{\mathbf{A}}} \mathrm{tr}(\tilde{\mathbf{Z}}^\top \tilde{\mathbf{A}})$ s.t. $\tilde{\mathbf{A}} \in \mathcal{G}$ {Find worst violator}
 6:    **if** $|\mathrm{tr}(\tilde{\mathbf{Z}}^\top \tilde{\mathbf{A}}) - \mathrm{tr}(\tilde{\mathbf{Z}}^\top \mathbf{A})| \geq \kappa$ **then**
 7:       add constraint to $\mathcal{K}$ : $\mathrm{tr}(\mathbf{Z}^\top \mathbf{A}) - \mathrm{tr}(\mathbf{Z}^\top \tilde{\mathbf{A}}) > 1 - \xi$
 8:    **end if**
 9: **until** $|\mathrm{tr}(\tilde{\mathbf{Z}}^\top \tilde{\mathbf{A}}) - \mathrm{tr}(\tilde{\mathbf{Z}}^\top \mathbf{A})| \leq \kappa$
10: **return** $\tilde{\mathbf{M}}$

---

Unfortunately, for networks larger than a few hundred nodes or for high-dimensional features, these SDPs do not scale adequately. The complexity of the SDP scales with the number of variables and constraints, yielding a worst-case time of $O(d^3 + \mathcal{C}^3)$ where $\mathcal{C} = O(n^2)$. By temporarily omitting the PSD requirement on $\mathbf{M}$, Algorithm 2 becomes equivalent to a one-class *structural support vector machine* (structural SVM). Stochastic SVM algorithms have been recently developed that have convergence time with no dependence on input size [19]. Therefore, we develop a large-scale algorithm based on projected stochastic subgradient descent. The proposed adaptation removes the dependence on $n$, where each iteration of the algorithm is $O(d^2)$, sampling one random constraint at a time. We can rewrite the optimization as unconstrained over an objective function with a hinge-loss on the structure preserving constraints:

$$f(\mathbf{M}) = \frac{\lambda}{2}||\mathbf{M}||_{\mathrm{F}}^2 - \frac{1}{|S|}\sum_{(i,j,k)\in S} \max(D_{\mathbf{M}}(\mathbf{x}_i, \mathbf{x}_j) - D_{\mathbf{M}}(\mathbf{x}_i, \mathbf{x}_k) + 1, 0).$$

Here the constraints have been written in terms of hinge-losses over triplets, each consisting of a node, its neighbor and its non-neighbor. The set of all such triplets is $S = \{(i,j,k) \mid A_{ij} = 1, A_{ik} = 0\}$. Using the distance transformation in Equation 1, each of the $|S|$ constraints can be written using a sparse matrix $\mathbf{C}^{(i,j,k)}$, where

$$C_{jj}^{(i,j,k)} = 1, \; C_{ik}^{(i,j,k)} = 1, \; ,C_{ki}^{(i,j,k)} = 1, \; ,C_{ij}^{(i,j,k)} = -1, \; C_{ji}^{(i,j,k)} = -1, \; ,C_{kk}^{(i,j,k)} = -1,$$

and whose other entries are zero. By construction, sparse matrix multiplication of $\mathbf{C}^{(i,j,k)}$ indexes the proper elements related to nodes $i$, $j$, and $k$, such that $\mathrm{tr}(\mathbf{C}^{(i,j,k)}\mathbf{X}^\top\mathbf{M}\mathbf{X})$ is equal to $D_{\mathbf{M}}(\mathbf{x}_i, \mathbf{x}_j) - D_{\mathbf{M}}(\mathbf{x}_i, \mathbf{x}_k)$. The subgradient of $f$ at $\mathbf{M}$ is then

$$\nabla f = \lambda\mathbf{M} + \frac{1}{|S|}\sum_{(i,j,k)\in S_+} \mathbf{X}\mathbf{C}^{(i,j,k)}\mathbf{X}^\top,$$

where $S_+ = \{(i,j,k)|D_{\mathbf{M}}(\mathbf{x}_i, \mathbf{x}_j) - D_{\mathbf{M}}(\mathbf{x}_i, \mathbf{x}_k) + 1 > 0\}$. If for all triplets this quantity is negative, there exists no unconnected neighbor of a point which is closer than a point's farthest connected neighbor – precisely the structure preserving criterion for nearest neighbor algorithms. In practice, we optimize this objective function via stochastic subgradient descent. We sample a batch of triplets, replacing $S$ in the objective function with a random subset of $S$ of size $B$. If a true metric is necessary, we intermittently project $\mathbf{M}$ onto the PSD cone. Full details about constructing the constraint matrices and minimizing the objective are shown in Algorithm 3.

---

**Algorithm 3** Structure preserving metric learning with nearest neighbor constraints and optimization with projected stochastic subgradient descent

---

**Input:** $\mathbf{A} \in \mathbb{B}^{n \times n}$, $\mathbf{X} \in \mathbb{R}^{d \times n}$, and parameters $\lambda, T, B$
1: $\mathbf{M}_1 \leftarrow \mathbf{I}_d$
2: **for** $t$ from 1 to $T - 1$ **do**
3:     $\eta_t \leftarrow \frac{1}{\lambda t}$
4:     $\mathbf{C} \leftarrow \mathbf{0}_{n,n}$
5:     **for** $b$ from 1 to $B$ **do**
6:         $(i,j,k) \leftarrow$ Sample random triplet from $S = \{(i,j,k) \mid A_{ij} = 1, A_{ik} = 0\}$
7:         **if** $D_{\mathbf{M}_t}(\mathbf{x}_i, \mathbf{x}_j) - D_{\mathbf{M}_t}(\mathbf{x}_i, \mathbf{x}_k) + 1 > 0$ **then**
8:            $\mathbf{C}_{jj} \leftarrow \mathbf{C}_{jj} + 1, \mathbf{C}_{ik} \leftarrow \mathbf{C}_{ik} + 1, \mathbf{C}_{ki} \leftarrow \mathbf{C}_{ki} + 1$
9:            $\mathbf{C}_{ij} \leftarrow \mathbf{C}_{ij} - 1, \mathbf{C}_{ji} \leftarrow \mathbf{C}_{ji} - 1, \mathbf{C}_{kk} \leftarrow \mathbf{C}_{kk} - 1$
10:        **end if**
11:    **end for**
12:     $\nabla_t \leftarrow \mathbf{X}\mathbf{C}\mathbf{X}^\top + \lambda\mathbf{M}_t$
13:     $\mathbf{M}_{t+1} \leftarrow \mathbf{M}_t - \eta_t\nabla_t$
14:     Optional: $\mathbf{M}_{t+1} \leftarrow [\mathbf{M}_{t+1}]^+$ {Project onto the PSD cone}
15: **end for**
16: **return** $\mathbf{M}_T$

---

## 2.3 Analysis

In this section, we provide analysis for the scaling behavior of SPML using SGD. A primary insight is that, since Algorithm 3 regularizes with the $L_2$ norm and penalizes with hinge-loss, omitting the

positive semidefinite requirement for $\mathbf{M}$ and vectorizing $\mathbf{M}$ makes the algorithm equivalent to a one-class, linear support vector machine with $O(n^3)$ input vectors. Thus, the stochastic optimization is an instance of the PEGAGOS algorithm [19], albeit a cleverly constructed one. The running time of PEGASOS does not depend on the input size, and instead only scales with the dimensionality, the desired optimization error on the objective function $\epsilon$ and the regularization parameter $\lambda$. The optimization error $\epsilon$ is defined as the difference between the found objective value and the true optimal objective value, $f(\tilde{\mathbf{M}}) - \min_{\mathbf{M}} f(\mathbf{M})$.

**Theorem 2** *Assume that the data is bounded such that* $\max_{(i,j,k) \in S} ||\mathbf{X}\mathbf{C}^{(i,j,k)}\mathbf{X}^\top||_{\mathrm{F}}^2 \leq R$, *and* $R \geq 1$. *During Algorithm 3 at iteration $T$, with $\lambda \leq 1/4$, and batch-size $B = 1$, let $\bar{\mathbf{M}} = \frac{1}{T}\sum_{t=1}^{T}\mathbf{M}_t$ be the average $\mathbf{M}$ so far. Then, with probability of at least $1 - \delta$,*

$$f(\bar{\mathbf{M}}) - \min_{\mathbf{M}} f(\mathbf{M}) \leq \frac{84R^2 \ln(T/\delta)}{\lambda T}.$$

*Consequently, the number of iterations necessary to reach an optimization error of $\epsilon$ is $\tilde{O}(\frac{1}{\lambda \epsilon})$.*

**Proof** The theorem is proven by realizing that Algorithm 3 is an instance of PEGASOS without a projection step on one-class data, since Corollary 2 in [20] proves this same bound for traditional SVM input, also without a projection step. The input to the SVM is the set of all $d \times d$ matrices $XC^{(i,j,k)}X^\top$ for each triplet $(i,j,k) \in S$. ∎

Note that the large size of set $S$ plays no role in the running time; each iteration requires $O(d^2)$ work. Assuming the node feature vectors are of bounded norm, the radius of the input data $R$ is constant with respect to $n$, since each is constructed using the feature vectors of three nodes. In practice, as in the PEGASOS algorithm, we propose using $\mathbf{M}_T$ as the output instead of the average, as doing so performs better on real data, but an averaging version is easily implemented by storing a running sum of $\mathbf{M}$ matrices and dividing by $T$ before returning.

Figure 2(b) shows the training and testing prediction performance on one of the experiments described in detail in Section 3 as stochastic SPML converges. The area under the receiver operator characteristic (ROC) curve is measured, which is related to the structure preserving hinge loss, and the plot clearly shows fast convergence and quickly diminishing returns at higher iteration counts.

## 2.4 Variations

While stochastic SPML does not scale with the size of the input graph, evaluating distances using a full $\mathbf{M}$ matrix requires $O(d^2)$ work. Thus, for high-dimensional data, one approach is to use *principal component analysis* or *random projections* to first reduce dimensionality. It has been shown that $n$ points can be mapped into a space of dimensionality $O(\log n / \varepsilon^2)$ such that distances are distorted by no more than a factor of $(1 \pm \varepsilon)$ [5, 11]. Another approach is to to limit $\mathbf{M}$ to be nonzero only along the diagonal. Diagonalizing $\mathbf{M}$ reduces the amount of work to $O(d)$.

If modeling cross-feature interactions is necessary, another option for reducing the computational cost is to perform SPML using a low-rank factorization of $\mathbf{M}$. In this case, all references to $\mathbf{M}$ can be replaced with $\mathbf{L}^\top \mathbf{L}$, thus inducing a true metric without projection. The updated gradient with respect to $\mathbf{L}$ is simply $\nabla_t \leftarrow 2\mathbf{X}\mathbf{C}\mathbf{X}^\top\mathbf{L}^\top + \lambda\mathbf{L}_t$. Using a factorization also allows replacing the regularizer with the Frobenius norm of the $\mathbf{L}$ matrix, which is equivalent to the *nuclear norm* of $\mathbf{M}$ [18]. Using this formulation causes the objective to no longer be convex, but seems to work well in practice. Finally, when predicting links of new nodes, SPML does not know how many connections to predict. To address this uncertainty, we propose a variant to SPML called *degree distributional metric learning* (DDML), which simultaneously learns the metric as well as parameters for the connectivity algorithm. Details on DDML and low-rank SPML are provided in the Appendix.

## 3 Experiments

We present a variety of synthetic and real-world experiments that elucidate the behavior of SPML. First we show how SPML performs on a simple synthetic dataset that is easily visualized in two

dimensions and which we believe mimics many traditional network datasets. We then demonstrate favorable performance for SPML in predicting links of the Wikipedia document network and the Facebook social network.

## 3.1 Synthetic example

To better understand the behavior of SPML, consider the following synthetic experiment. First $n$ points are sampled from a $d$-dimensional uniform distribution. These vectors represent the true features for the $n$ nodes $\mathbf{X} \in \mathbb{R}^{d \times n}$. We then compute an adjacency matrix by performing a minimum-distance $b$-matching on $\mathbf{X}$. Next, the true features are scrambled by applying a random linear transformation: $\mathbf{RX}$ where $\mathbf{R} \in \mathbb{R}^{d \times d}$. Given $\mathbf{RX}$ and $\mathbf{A}$, the goal of SPML is to learn a metric $\mathbf{M}$ that undoes the linear scrambling, so that when $b$-matching is applied to $\mathbf{RX}$ using the learned distance metric, it produces the input adjacency matrix.

Figure 1 illustrates the results of the above experiment for $d = 2$, $n = 50$, and $b = 4$. In Figure 1(a), we see an embedding of the graph using the true features for each node as coordinates, and connectivity generated from $b$-matching. In Figure 1(b), the random linear transformation has been applied. We posit that many real-world datasets resemble plot 1(b), with seemingly incongruous feature and connectivity information. Applying $b$-matching to the scrambled data produces connections shown in Figure 1(c). Finally, by learning $\mathbf{M}$ via SPML (Algorithm 2) and computing $\mathbf{L}$ by Cholesky decomposition of $\mathbf{M}$, we can recover features $\mathbf{LRX}$ (Figure 1(d)) that respect the structure in the target adjacency matrix and thus more closely resemble the true features used to generate the data.

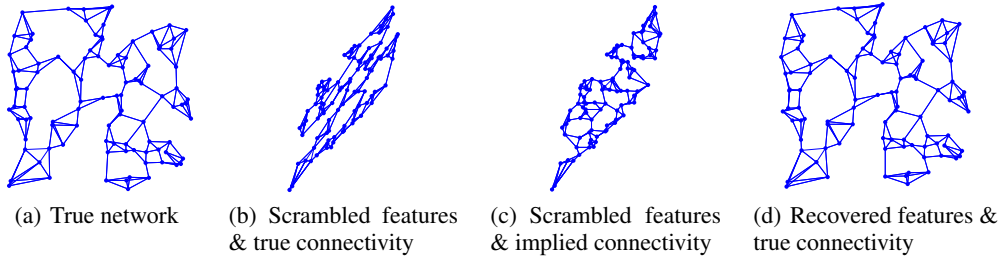

(a) True network     (b) Scrambled features & true connectivity     (c) Scrambled features & implied connectivity     (d) Recovered features & true connectivity

Figure 1: In this synthetic experiment, SPML finds a metric that inverts the random transformation applied to the features (b), such that under the learned metric (d) the implied connectivity is identical to the original connectivity (a) as opposed to inducing a different connectivity (c).

## 3.2 Link prediction

We compare SPML to a variety of methods for predicting links from node features: Euclidean distances, *relational topic models* (RTM) , and traditional *support vector machines* (SVM). A simple baseline for comparison is how well the Euclidean distance metric performs at ranking possible connections. Relational topic models learn a link probability function in addition to latent topic mixtures describing each node [2]. For the SVM, we construct training examples consisting of the pairwise differences between node features. Training examples are labeled positive if there exists an edge between the corresponding pair of nodes, and negative if there is no edge. Because there are potentially $O(n^2)$ possible examples, and the graphs are sparse, we subsample the negative examples so that we include a randomly chosen equal number of negative examples as positive edges. Without subsampling, the SVM is unable to run our experiments in a reasonable time. We use the SVMPerf implementation for our SVM [9], and the authors' code for RTM [2].

Interestingly, an SVM with these inputs can be interpreted as an instance of SPML using diagonal $\mathbf{M}$ and the $\epsilon$-neighborhood connectivity algorithm, which connects points based on their distance, completely independently of the rest of the graph structure. We thus expect to see better performance using SPML in cases where the structure is important. The RTM approach is appropriate for data that consists of counts, and is a generative model which recovers a set of topics in addition to link predictions. Despite the generality of the model, RTM does not seem to perform as well as discriminative methods in our experiments, especially in the Facebook experiment where the data is quite different from bag-of-words features. For SPML, we run the stochastic algorithm with batch size 10. We skip the PSD projection step, since these experiments are only concerned with

prediction, and obtaining a true metric is not necessary. SPML is implemented in MATLAB and requires only a few minutes to converge for each of the experiments below.

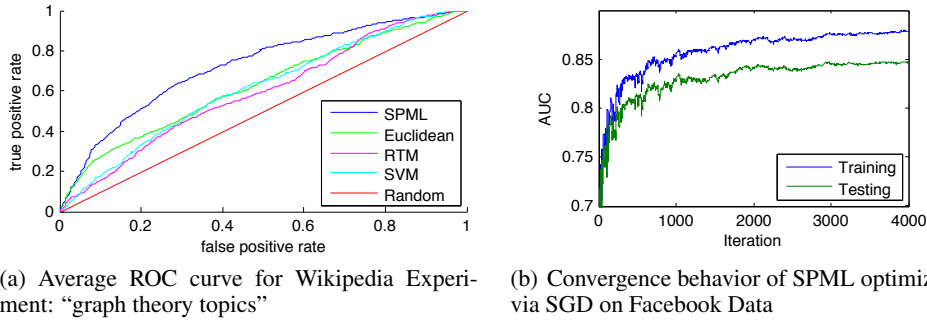

(a) Average ROC curve for Wikipedia Experiment: "graph theory topics"

(b) Convergence behavior of SPML optimized via SGD on Facebook Data

Figure 2: Average ROC performance for the "graph theory topics" Wikipedia experiment (left) shows a strong lift for SPML over competing methods. We see that SPML converges quickly with diminishing returns after many iterations (right).

**Wikipedia articles**   We apply SPML to predicting links on Wikipedia pages. Imagine the scenario where an author writes a new Wikipedia entry and then, by analyzing the word counts on the newly written page, an algorithm is able to suggest which other Wikipedia pages it should link to. We first create a few subnetworks consisting of all the pages in a given category, their bag-of-words features, and their connections. We choose three categories: "graph theory topics", "philosophy concepts", and "search engines". We use a word dictionary of common words with stop-words removed. For each network, we split the data 80/20 for training and testing, where 20% of the nodes are held out for evaluation. On the remaining 80% we cross-validate (five folds) over the parameters for each algorithm (RTM, SVM, SPML), and train a model using the best-scoring regularization parameter. For SPML, we use the diagonal variant of Algorithm 3, since the high-dimensionality of the input features reduces the benefit of cross-feature weights. On the held-out nodes, we task each algorithm to rank the unknown edges according to distance (or another measure of link likelihood), and compare the accuracy of the rankings using *receiver operator characteristic* (ROC) curves. Table 1 lists the statistics of each category and the average area under the curve (AUC) over three train/test splits for each algorithm. A ROC curve for the "graph theory" category is shown in Figure 2(a). For "graph theory" and "search engines", SPML provides a distinct advantage over other methods, while no method has a particular advantage on "philosophy concepts". One possible explanation for why the SVM is unable to gain performance over Euclidean distance is that the wide range of degrees for nodes in these graphs makes it difficult to find a single threshold that separates edges from non-edges. In particular, the "search engines" category had an extremely skewed degree distribution, and is where SPML shows the greatest improvement.

We also apply SPML to a larger subset of the Wikipedia network, by collecting word counts and connections of 100,000 articles in a breadth-first search rooted at the article "Philosophy". The experimental setup is the same as previous experiments, but we use a $0.5\%$ sample of the nodes for testing. The final training algorithm ran for 50,000 iterations, taking approximately ten minutes on a desktop computer. The resulting AUC on the edges of the held-out nodes is listed in Table 1 as the "Philosophy Crawl" dataset. The SVM and RTM do not scale to data of this size, whereas SPML offers a clear advantage over using Euclidean distance for predicting links.

**Facebook social networks**   Applying SPML to social network data allows us to more accurately predict who will become friends based on the profile information for those users. We use Facebook data [22], where we have a small subset of anonymized profile information for each student of a university, as well as friendship information. The profile information consists of gender, status (meaning student, staff, or faculty), dorm, major, and class year. Similarly to the Wikipedia experiments in the previous section, we compared SPML to Euclidean, RTM, and SVM. For SPML, we learn a full $\mathbf{M}$ via Algorithm 3. For each person, we construct a sparse feature vector where there is one feature corresponding to every possible dorm, major, etc. for each feature type. We select only people who have indicated all five feature types on their profiles. Table 1 shows details of

Table 1: Wikipedia (top), Facebook (bottom) dataset and experiment information. Shown below: number of nodes $n$, number of edges $m$, dimensionality $d$, and AUC performance.

|  | $n$ | $m$ | $d$ | Euclidean | RTM | SVM | SPML |
|---|---|---|---|---|---|---|---|
| Graph Theory | 223 | 917 | 6695 | 0.624 | 0.591 | 0.610 | **0.722** |
| Philosophy Concepts | 303 | 921 | 6695 | 0.705 | 0.571 | **0.708** | 0.707 |
| Search Engines | 269 | 332 | 6695 | 0.662 | 0.487 | 0.611 | **0.742** |
| Philosophy Crawl | 100,000 | 4,489,166 | 7702 | 0.547 | – | – | **0.601** |
| Harvard | 1937 | 48,980 | 193 | 0.764 | 0.562 | 0.839 | **0.854** |
| MIT | 2128 | 95,322 | 173 | 0.702 | 0.494 | 0.784 | **0.801** |
| Stanford | 3014 | 147,516 | 270 | 0.718 | 0.532 | 0.784 | **0.808** |
| Columbia | 3050 | 118,838 | 251 | 0.717 | 0.519 | 0.796 | **0.818** |

the Facebook networks for the four schools we consider: Harvard, MIT, Stanford, and Columbia. We perform a separate experiment for each school, randomly splitting the data 80/20 for training and testing. We use the training data to select parameters via five-fold cross validation, and train a model. The AUC performance on the held-out edges is also listed in Table 1. It is clear from the quantitative results that structural information is contributing to higher performance for SPML as compared to other methods.

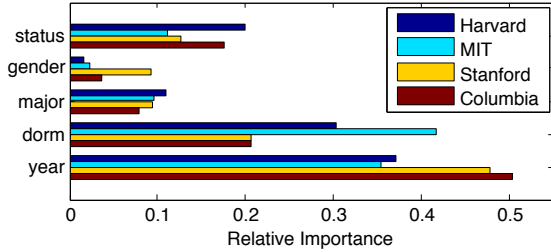

Figure 3: Comparison of Facebook social networks from four schools in terms of feature importance computed from the learned structure preserving metric.

By looking at the weight of the diagonal values in $\mathbf{M}$ normalized by the total weight, we can determine which feature differences are most important for determining connectivity. Figure 3 shows the normalized weights averaged by feature types for Facebook data. Here we see the feature types compared across four schools. For all schools except MIT, the graduating year is most important for determining distance between people. For MIT, dorms are the most important features. A possible explanation for this difference is that MIT is the only school in the list that makes it easy for students to stay in a residence for all four years of their undergraduate program, and therefore which dorm one lives in may affect more strongly the people they connect to.

## 4 Discussion

We have demonstrated a fast convex optimization for learning a distance metric from a network such that the distances are tied to the network's inherent topological structure. The structure preserving distance metrics introduced in this article allow us to better model and predict the behavior of large real-world networks. Furthermore, these metrics are as lightweight as independent pairwise models, but capture structural dependency from features making them easy to use in practice for link-prediction. In future work, we plan to exploit SPML's lack of dependence on graph size to learn a structure preserving metric on massive-scale graphs, e.g., the entire Wikipedia site. Since each iteration requires only sampling a random node, following a link to a neighbor, and sampling a non-neighbor, this can all be done in an online fashion as the algorithm crawls a network such as the worldwide web, learning a metric that may gradually change over time.

**Acknowledgments** This material is based upon work supported by the National Science Foundation under Grant No. 1117631, by a Google Research Award, and by the Department of Homeland Security under Grant No. N66001-09-C-0080.

## Footnotes

[1]In the remainder of the paper, we interchangeably use $\mathcal{G}$ to denote the set of feasible graphs and the algorithm used to find the optimal connectivity within the set of feasible graphs.

# References

[1] E. Airoldi, D. Blei, S. Fienberg, and E. Xing. Mixed membership stochastic blockmodels. *JMLR*, 9:1981–2014, 2008.

[2] J. Chang and D. Blei. Hierarchical relational models for document networks. *Annals of Applied Statistics*, 4:124–150, 2010.

[3] G. Chechik, V. Sharma, U. Shalit, and S. Bengio. Large scale online learning of image similarity through ranking. *J. Mach. Learn. Res.*, 11:1109–1135, March 2010.

[4] J. Chen, W. Geyer, C. Dugan, M. Muller, and I. Guy. Make new friends, but keep the old: recommending people on social networking sites. In *CHI*, pages 201–210. ACM, 2009.

[5] S. Dasgupta and A. Gupta. An elementary proof of a theorem of Johnson and Lindenstrauss. *Random Struct. Algorithms*, 22:60–65, January 2003.

[6] C. Fremuth-Paeger and D. Jungnickel. Balanced network flows, a unifying framework for design and analysis of matching algorithms. *Networks*, 33(1):1–28, 1999.

[7] B. Huang and T. Jebara. Loopy belief propagation for bipartite maximum weight b-matching. In *Proceedings of the Eleventh International Conference on Artificial Intelligence and Statistics*, volume 2 of JMLR: W&CP, pages 195–202, 2007.

[8] B. Huang and T. Jebara. Fast b-matching via sufficient selection belief propagation. In *Proceedings of the Fourteenth International Conference on Artificial Intelligence and Statistics*, 2011.

[9] T. Joachims. Training linear SVMs in linear time. In *ACM SIG International Conference On Knowledge Discovery and Data Mining (KDD)*, pages 217 – 226, 2006.

[10] T. Joachims, T. Finley, and C. Yu. Cutting-plane training of structural SVMs. *Machine Learning*, 77(1):27–59, 2009.

[11] W. Johnson and J. Lindenstrauss. Extensions of Lipschitz maps into a Hilbert space. *Contemporary Mathematics*, (26):189–206, 1984.

[12] J. Leskovec and E. Horvitz. Planetary-scale views on a large instant-messaging network. *ACM WWW*, 2008.

[13] J. Leskovec, J Kleinberg, and C. Faloutsos. Graphs over time: densification laws, shrinking diameters and possible explanations. In *Proc. of the Eleventh ACM SIGKDD International Conference on Knowledge Discovery in Data Mining*, 2005.

[14] M. Middendorf, E. Ziv, C. Adams, J. Hom, R. Koytcheff, C. Levovitz, and G. Woods. Discriminative topological features reveal biological network mechanisms. *BMC Bioinformatics*, 5:1471–2105, 2004.

[15] G. Namata, H. Sharara, and L. Getoor. A survey of link mining tasks for analyzing noisy and incomplete networks. In *Link Mining: Models, Algorithms, and Applications*. Springer, 2010.

[16] M. Newman. The structure and function of complex networks. *SIAM REVIEW*, 45:167–256, 2003.

[17] M. Newman. Analysis of weighted networks. *Phys. Rev. E*, 70(5):056131, Nov 2004.

[18] J. Rennie and N. Srebro. Fast maximum margin matrix factorization for collaborative prediction. In *Proceedings of the Twenty-Second International Conference*, volume 119 of *ACM International Conference Proceeding Series*, pages 713–719. ACM, 2005.

[19] S. Shalev-Shwartz, Y. Singer, and N. Srebro. Pegasos: Primal estimated sub-gradient solver for SVM. In *Proceedings of the 24th International Conference on Machine Learning*, ICML '07, pages 807–814, New York, NY, USA, 2007. ACM.

[20] S. Shalev-Shwartz, Y. Singer, N. Srebro, and A. Cotter. Pegasos: Primal estimated sub-gradient solver for SVM. *Mathematical Programming*, To appear.

[21] B. Shaw and T. Jebara. Structure preserving embedding. In *Proc. of the $26^{th}$ International Conference on Machine Learning*, 2009.

[22] A. Traud, P. Mucha, and M. Porter. Social structure of Facebook networks. *CoRR*, abs/1102.2166, 2011.

[23] K. Weinberger and L. Saul. Distance metric learning for large margin nearest neighbor classification. *Journal of Machine Learning Research*, 10:207–244, 2009.

[24] E. Xing, A. Ng, M. Jordan, and S. Russell. Distance metric learning with application to clustering with side-information. In S. Becker, S. Thrun, and K. Obermayer, editors, *NIPS*, pages 505–512. MIT Press, 2002.

[25] J. Xu and Y. Li. Discovering disease-genes by topological features in human protein-protein interaction network. *Bioinformatics*, 22(22):2800–2805, 2006.

[26] T. Yang, R. Jin, Y. Chi, and S. Zhu. Combining link and content for community detection: a discriminative approach. In *Proceedings of the 15th ACM SIGKDD international conference on Knowledge discovery and data mining*, KDD '09, pages 927–936, New York, NY, USA, 2009. ACM.

